# A Simple yet Scalable Granger Causal Structural Learning Approach for Topological Event Sequences

**Mingjia Li**[*]  **Shuo Liu**[*]  **Hong Qian**[†]  **Aimin Zhou**
Shanghai Institute of AI for Education and School of Computer Science and Technology,
East China Normal University, Shanghai 200062, China
{limj,shuoliu}@stu.ecnu.edu.cn
{hqian,amzhou}@cs.ecnu.edu.cn

## Abstract

In modern telecommunication networks, faults manifest as alarms, generating thousands of events daily. Network operators need an efficient method to identify the root causes of these alarms to mitigate potential losses. This task is challenging due to the increasing scale of telecommunication networks and the interconnected nature of devices, where one fault can trigger a cascade of alarms across multiple devices within a topological network. Recent years have seen a growing focus on causal approaches to addressing this problem, emphasizing the importance of learning a Granger causal graph from topological event sequences. Such causal graphs delineate the relations among alarms and can significantly aid engineers in identifying and rectifying faults. However, existing methods either ignore the topological relationships among devices or suffer from relatively low scalability and efficiency, failing to deliver high-quality responses in a timely manner. To this end, this paper proposes S²GCSL, a simple yet scalable Granger causal structural learning approach for topological event sequences. S²GCSL utilizes a linear kernel to model activation interactions among various event types within a topological network, and employs gradient descent to efficiently optimize the likelihood function. Notably, it can seamlessly incorporate expert knowledge as constraints within the optimization process, which enhances the interpretability of the outcomes. Extensive experimental results on both large-scale synthetic and real-world problems verify the scalability and efficacy of S²GCSL.

## 1 Introduction

Telecommunication networks are an important component of infrastructure of modern society, where thousands of alarms may be generated by various types of faults on a daily basis. It is crucial for network operators to efficiently identify the root causes of these alarms since for every additional minute that a fault persists in telecommunication networks, it can lead to significant economic losses and cause negative public sentiment. However, the increasing scale of telecommunication networks and the interconnected nature of devices make this task particularly challenging. A single fault has the potential to trigger a cascade of alarms across multiple devices within a topological network. How to model the generation of event sequences, especially in large-scale, multi-event-type scenarios, has become more and more urgent in the telecommunication network fault diagnosis (TNFD) task.

One promising approach to understanding and predicting the generation and propagation of event sequences is the application of Granger causality analysis [Granger, 1969], which in the context of

---

[*]Equal contribution.
[†]Hong Qian is the corresponding author.

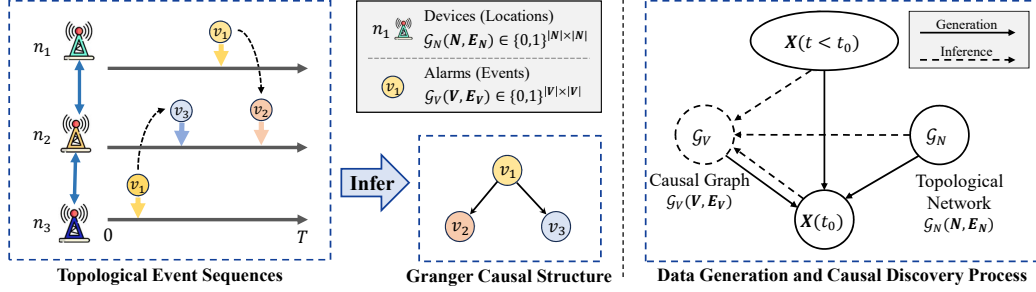

Figure 1: The left panel shows an example of the topological event sequences generated by a mobile network, where a Granger causal graph is expected to be inferred to serve the TNFD task. The right panel illustrates the abstracted generation and inference processes. Solid lines represent the data generation process for event sequences, while dashed lines correspond to the inference process for the causal graph. Solid circles denote observed variables, and dashed circles represent latent variables.

TNFD, can help in identifying whether a type of alarm can be the evidence of the occurrence of particular faults. As shown in the left panel of Figure 1, the mutual activation effects among various types of events are considered as a form of Granger causality and the task can be formulated as a causal structural learning problem. However, within the scope of learning Granger causality from event sequences, most of the methods rely on the assumption that event sequences are independent and identically distributed (i.i.d.). Yet, in the scenario of TNFD, the topological nature of devices inherently links events across the network, challenging the validity of such i.i.d. assumption.

Fortunately, a number of notable methods [Cai et al., 2024; Liu et al., 2024] have emerged very recently to solve the problem of Granger causal structural learning from event sequences under non-i.i.d. assumptions, where the generation of alarm sequences is modeled by the topological multivariate Hawkes process (TMHP) [Cai et al., 2024]. However, all of these existing TMHP-based approaches suffer from the issue of efficiency or scalability, which is essential in the scenario of large-scale TNFD. Specifically, topological Hawkes process (THP) [Cai et al., 2024] proposes the TMHP model whose likelihood function of the Granger causal graph is optimized through a gradient-free manner, where the optimization of structure and parameter is decoupled, leading to relative inefficiency. On the other hand, the topological neural Poisson auto-regressive model (TNPAR) [Liu et al., 2024] proposes to utilize the neural point process (NPP) to implement the intensity function in TMHP, which greatly enhances the model's ability to model complex relationships among events. However, since NPP relies on deep neural networks, it lacks an analytical expression for its likelihood function. Consequently, optimization is only feasible through the prediction of event sequences, which slows down the training process and limits the model's scalability.

To address the challenges of scalability as well as efficiency presented in the TNFD task, this paper proposes S$^2$GCSL, a simple yet scalable Granger causal structural learning approach for topological event sequences. S$^2$GCSL utilizes a linear kernel to implement the activation effect among various event types and adopt a gradient descent manner to optimize the likelihood function. It is worth noting that multiple prior expert knowledge, namely, sparsity and acyclicity, can be integrated into this optimization procedure in a simple form of constraints, which improves the interpretability of results. Extensive experiments on both large-scale synthetic problems and a real-world TNFD problem on metropolitan telecommunication network alarm data verify the scalability and efficacy of S$^2$GCSL. In a nutshell, the main contribution of this paper includes:

- We propose a simple yet scalable Granger causal structural learning method, incorporating simple modeling and gradient-based optimization, to efficiently infer Granger causal graph from topological event sequences of alarms in telecommunication networks.

- By incorporating experts' prior knowledge as constraints into the objective function, we provide a simple method to guarantee the interpretability during the process of optimization within the context of TNFD.

- Extensive experiments show that S$^2$GCSL not only achieves superior performance in effectiveness, efficiency and scalability, but also maintains robustness across diverse scenarios, validating its practical applicability in real-world telecommunication network environments.

The subsequent sections of this paper respectively recap the related work, introduce the proposed method, present the experimental results and analysis, and finally conclude the paper.

## 2  Related Work

**Temporal Point Processes.** Temporal point processes are stochastic processes used for modeling event sequences. They can be categorized into statistical point processes and neural point processes. Statistical point processes focus on developing appropriate intensity functions, often with parameters that have specific physical interpretations. Examples of statistical point processes include the Poisson process [Cox and R, 1955], Hawkes process [Hawkes and G, 1971], self-correcting process [Isham et al., 1979] and reactive point process [Ertekin et al., 2015]. On the other hand, NPP [Shchur et al., 2021] utilize the powerful learning capabilities of neural networks to implement the intensity functions, allowing the model to potentially capture complex relationships among events.

**Granger Causality for Event Sequences.** Various methods exist for discovering Granger causality from event sequences. For example, Hawkes process based methods [Zhou et al., 2013; Xu et al., 2016] operate on the assumption that past events stimulate the occurrence of related events in the future only if the former Granger causes the latter. However, these Hawkes process based methods significantly rely on the i.i.d. assumption, which is violated in TNFD due to the interconnected nature of devices. To address the non-i.i.d. challenge, THP [Cai et al., 2024] generalizes the Hawkes process to non-i.i.d. case by incorporating the topological information among devices. However, due to the use of gradient-free optimization to search for Granger causal structures, THP has efficiency shortcomings, which could undermine its competitiveness in the TNFD task. Inspired by THP, TNPAR [Liu et al., 2024] further proposes to utilize an NPP to model the intensity function, which significantly improves the model's capability to represent intricate event relationships. However, due to the lack of an analytical expression for the likelihood function, NPP optimization is limited to reconstructing training data from event sequences and conducting training in a supervised learning manner, which not only lacks theoretical guarantees but also performs relatively inefficient. This could make TNPAR unable to handle large-scale problems as well. Other NPP based methods include causality from attributions on sequence of events (CAUSE) [Zhang et al., 2020] and transformer Hawkes process (TransHP) [Zuo et al., 2020]. CAUSE uses an attribution method to extract Granger causality from the well-trained NPP, and in TransHP, the temporal dependencies among event sequences are captured by a transformer. These NPP based methods are typically flexible, and thus can incorporate topology information through some modifications. Unfortunately, the scalability of these methods are also not ideal due to the same reason as TNPAR.

In a different context, our work is also related to Granger causal discovery from time series. In [Brillinger and R, 1994], researchers aggregate event sequences into time series, enabling the analysis of event sequences using auto-regressive models. Amortized causal discovery (ACD) [Löwe et al., 2022] applies an amortized model to infer Granger causal structures from time series. In addition, PC with momentary conditional independence test (PCMCI) [Runge, 2020] and methods based on transfer entropy [Mijatovic et al., 2021; Chen et al., 2020], are founded on strict causal assumptions rather than Granger causality and utilize independence tests or measures to discover the causal structure from event sequences.

## 3  The Proposed S$^2$GCSL

### 3.1  Problem Formulation

In the problem of learning Granger causal structure from topological event sequences in telecommunication networks, suppose the topological connections between devices are represented as an undirected graph $\mathcal{G}_N(\boldsymbol{N}, \boldsymbol{E_N})$, where $\boldsymbol{N}$ is the device set and the edges $E_N$ indicate physical connections between these devices. Besides, a directed graph $\mathcal{G}_V(\boldsymbol{V}, \boldsymbol{E_V})$ captures the Granger causal relationships among different event types $\boldsymbol{V}$, with $\boldsymbol{E_V}$ representing the causal edges between two different event types. In this scenario, an event can not only influence future events at its location but also at devices that are topologically connected to it.

Given this setup, we consider topological event sequences of length $m$, denoted by $\boldsymbol{X} = \{(v_i, n_i, t_i, \ell_i) | i \in \{1, \ldots, m\}\}$. These sequences arise from the causal interactions defined in

$\mathcal{G}_V$ and occur within the structure of $\mathcal{G}_N$. Here, $v_i \in \boldsymbol{V}$ represents the event type, $n_i \in \boldsymbol{N}$ denotes the device where the event occurs, $t_i \in [0, T]$ refer to the start timestamps of the event and $\ell_i \in \mathbb{Z}^+$ denotes how many timestamps the event last. We can model the occurrence of these events as a series of counting processes $\{C_{v,n}(t) | v \in \boldsymbol{V}, n \in \boldsymbol{N}, t \in [0, T]\}$, where $C_{v,n}(t)$ counts the number of times event type $v$ has occurred at device $n$ up to time $t$. For practical analysis, the continuous interval $[0, T]$ is segmented into $\lceil T/\Delta \rceil$ smaller intervals, where $\Delta \in \mathbb{R}^+$ is chosen based on the application context. The number of occurrences within these intervals are recorded as $\boldsymbol{O^{V,N}} = \{O_t^{v,n} | t \in \{1, \ldots, \lceil T/\Delta \rceil\}, v \in \boldsymbol{V}, n \in \boldsymbol{N}\}$, where $O_t^{v,n} = C_{v,n}(t \times \Delta) - C_{v,n}((t-1) \times \Delta)$ reflects the occurrences of event type $v$ on device $n$ within the interval $((t-1) \times \Delta, t \times \Delta]$.

Assuming that the counting processes within each time interval follow a Poisson process, where the intensity $\lambda_t^{v,n}$ refers to the expected number of occurrences in an interval of event type $v$ on device $n$ at time $t$, and the probability of observing $O_t^{v,n}$ occurrences can be formulated using the Poisson probability function as follows:

$$P(O_t^{v,n} = o) = \frac{(\lambda_t^{v,n} \Delta)^o}{o\,!} e^{-\lambda_t^{v,n} \Delta}, \quad o = 0, 1, 2, \ldots. \tag{1}$$

This sets the stage to define the problem of causal discovery from topological event sequences as Definition 1 below.

**Definition 1** (Granger Causal Discovery from Topological Event Sequences). *Given the records of event sequences $\boldsymbol{X} = \{(v_i, n_i, t_i, \ell_i) | i \in \{1, \ldots, m\}\}$, and the existing topological network $\mathcal{G}_N(\boldsymbol{N}, \boldsymbol{E_N})$, the objective of Granger causal discovery from topological event sequences is to deduce the Granger causal graph $\mathcal{G}_V(\boldsymbol{V}, \boldsymbol{E_V})$ among the event types.*

To address this problem, we introduce the S$^2$GCSL model, which includes both a generation process and an inference process.

## 3.2 Generation Process of the Event Sequences

The generation process of S$^2$GCSL is illustrated with solid lines in the right panel of Figure 1, where the occurrence $O_t^{v_i, n_j}$ is influenced by historical event sequences $\boldsymbol{X}_h = \{(v_i, n_i, t_i, \ell_i) | t_i < t \le t_i + \ell_i\}$, along with two types of matrices, $\boldsymbol{A}$ and $\boldsymbol{B}_{0:k}$. Here, $\boldsymbol{A}$ is a $|\boldsymbol{V}| \times |\boldsymbol{V}|$ binary matrix indicating Granger causality between event types. The element $\boldsymbol{A}_{v_i, v_j}$ from matrix $\boldsymbol{A}$ at row $i$ and column $j$ denotes the causality: if $\boldsymbol{A}_{v_i, v_j} = 0$, it means that there is no Granger causality between event type $v_i$ and $v_j$. Otherwise, there exists a causality. $\boldsymbol{B}_{0:k}$ represents a set of matrices $\{\boldsymbol{B}_0, \boldsymbol{B}_1, \ldots, \boldsymbol{B}_k\}$, where each $\boldsymbol{B}_k$ is a $|\boldsymbol{N}| \times |\boldsymbol{N}|$ binary matrix showing the physical connections between devices at a geodesic distance [Bouttier et al., 2003] of $k$ within $\mathcal{G}_N$. The element $\boldsymbol{B}_{n_i, n_j}^k$ in matrix $\boldsymbol{B}_k$ at row $i$ and column $j$ specifies this connection: if $\boldsymbol{B}_{n_i, n_j}^k = 1$, then the geodesic distance between device $n_i$ and $n_j$ is less than or equal to $k$. Otherwise, $\boldsymbol{B}_{n_i, n_j}^k$ is set to 0.

In this paper, we utilize a linear activation function to model the causal relationships among event types, and the intensity $\lambda_t^{v,n}$ of an event type $v$ on a device $n$ is given as:

$$\lambda_t^{v,n} = \mu^{v,n} + \sum_{i: t_i < t \le t_i + \ell_i} \alpha_{v_i v} \boldsymbol{A}_{v_i, v} \boldsymbol{B}_{n_i, n}^k, \tag{2}$$

where $\mu^{v,n}$ is a constant representing the baseline intensity of event type $v$ on device $n$ and $\alpha_{v_i v}$ is a coefficient denoting the activation intensity of event type $v_i$ on event type $v$, which can be considered as the Granger causality of $v_i \to v$. $\sum_{i: t_i < t \le t_i + \ell_i}$ means the summation over all the events $j$ that the event $i$ occurs during the duration of $j$.

Counterintuitively, such a simple assumption based on linear activation function can effectively model complex functions that depict how event intensities vary over time. This capability arises because the intensity of an event at a given time may be influenced by countless preceding events. Even though the impact of each individual event is simple, the superposition of numerous linear functions can approximate arbitrarily complex intensity functions.

## 3.3 Inference Process of the Granger Causal Graph

The inference mechanism of S²GCSL is illustrated with dashed lines in the right panel of Figure 1. The diagram indicates that the causal matrices $\boldsymbol{A}$ are deduced by integrating the current event occurrences $\boldsymbol{X}(t_0)$, historical event sequences $\boldsymbol{X}_h = \{(v_i, n_i, t_i, \ell_i)|t_i < t \leq t_i + \ell_i\}$, and the topological network $\boldsymbol{B}_{0:k}$.

Considering the generation model defined in Eq. 2, the coefficient $\alpha_{v_i v}$ is treated as the Granger causality statistic of event type $v_i$ on event type $v$. Consequently, the weighted Granger causal matrix $\boldsymbol{A'} \in \mathbb{R}^{|\boldsymbol{V}| \times |\boldsymbol{V}|}$, where $\boldsymbol{A'}_{i,j} = \alpha_{i,j} \times \boldsymbol{A}_{i,j}$, along with the baseline intensity matrix $\boldsymbol{\mu}$ can be seen as the parameters of the generation model $\Theta = \{\boldsymbol{A'}, \boldsymbol{\mu}\}$, and the log-likelihood function of $\Theta$ given the observed event sequences $\boldsymbol{X} = \{(v_i, n_i, t_i, \ell_i)|i \in \{1, \dots, m\}\}$ can be expressed as follows:

$$L(\boldsymbol{A'}, \boldsymbol{\mu}) = \sum_n \left( \sum_{i:n_i=n} \log \lambda_{t_i}^{v_i,n} - \sum_v \int_0^T \lambda_t^{v,n} \, dt \right)$$

$$= \sum_n \left( \sum_{i:n_i=n} \log(\mu^{v_i,n} + \sum_{j:t_j<t_i\leq t_j+\ell_j} \boldsymbol{A'}_{v_j,v_i} \boldsymbol{B}_{n_j,n_i}^k) \right.$$

$$\left. -T \sum_v \mu^{v,n} - \sum_v \sum_{i:n_i=n} \boldsymbol{A'}_{v_i,v} \boldsymbol{B}_{n_i,n}^k \ell_i) \right), \tag{3}$$

where $\sum_{i:n_i=n}$ means the summation over all the events $i$ that occur on device $n$. With such log-likelihood function, the Granger causal discovery problem can be transformed into the following optimization problem that estimates the optimal parameters $\Theta^\star = \{\boldsymbol{A^\star}, \boldsymbol{\mu^\star}\}$ as:

$$\boldsymbol{A^\star}, \boldsymbol{\mu^\star} = \underset{\boldsymbol{A'},\boldsymbol{\mu}}{\arg\min} - L(\boldsymbol{A'}, \boldsymbol{\mu}), \tag{4}$$

where $\boldsymbol{A^\star}$ denotes the inferred weighted Granger causal matrix.

As mentioned before, some prior expert knowledge of the causal structures in the TNFD scenario should be taken into consideration to ensure the inferred causal graph interpretable. Specifically, we focus on two properties of the causal graph among event types, namely sparsity and acyclicity. The sparsity of activation effects suggest that most event types only influence a small fraction of other event types in telecommunication networks. The sparsity can be reflected in the entry-norm constraint of $\boldsymbol{A'}$ as:

$$||\boldsymbol{A'}||_1 = \sum_{i=1}^{|\boldsymbol{V}|} \sum_{j=1}^{|\boldsymbol{V}|} |\boldsymbol{A'}_{i,j}| \leq \epsilon, \tag{5}$$

where $||\boldsymbol{A'}||_1$ denotes the $_{1,1}$ entry-norm of $\boldsymbol{A'}$ and $\epsilon$ is a small positive constant.

Besides, acyclicity suggests that the activation effects among event types should not form cycles and there is no self-excitation. As a result, the deduced causal graph should a DAG, which motivates us to introduce the acyclic constraint [Zheng et al., 2018] as:

$$h(\boldsymbol{A'}) = trace[(\boldsymbol{I} + \beta \boldsymbol{A'} \circ \boldsymbol{A'})^{|\boldsymbol{V}|}] - |V| = 0, \tag{6}$$

where $\boldsymbol{I}$ is the identity matrix, $\beta$ is a constant and $\circ$ denotes the Hardamard product. Consequently, the optimization with constraints can be written as:

$$\underset{\boldsymbol{A'},\boldsymbol{\mu}}{\min} - L(\boldsymbol{A'}, \boldsymbol{\mu}) \quad s.t. ||\boldsymbol{A'}||_1 \leq \epsilon, h(\boldsymbol{A'}) = 0. \tag{7}$$

By leveraging the Lagrangian multiplier method, the final optimization problem is defined as:

$$\boldsymbol{A^\star}, \boldsymbol{\mu^\star} = \underset{\boldsymbol{A'},\boldsymbol{\mu}}{\arg\min} - L(\boldsymbol{A'}, \boldsymbol{\mu}) + \lambda_1 ||\boldsymbol{A'}||_1 + \lambda_2 h(\boldsymbol{A'}), \tag{8}$$

where $\lambda_1$ and $\lambda_2$ refer to the regularized hyperparameters. For the optimization procedure, the Adam optimizer [Kingma and Ba, 2015] which is known for its efficiency and stability is adopted with learning rate $lr = 0.05$.

After the optimization, we got the weighted Granger causal matrix $\boldsymbol{A}^\star$, and to deduce the final binary adjancency matrix $\boldsymbol{A}$, edges with too small weights are pruned, i.e.,

$$\boldsymbol{A} = (a_{i,j}), \quad \text{where} \quad a_{i,j} = \begin{cases} 1 & \text{if } \boldsymbol{A}_{i,j}^\star > \rho \\ 0 & \text{if } \boldsymbol{A}_{i,j}^\star \le \rho \end{cases}, \tag{9}$$

and $\rho$ is a constant hyperparameter to control the pruning threshold.

## 4 Experiments

In this section, we will implement the proposed S$^2$GCSL method and the baseline approaches on both simulated data and metropolitan telecommunication network alarm data. All methods will undergo testing using 10 different random seeds, and the results will be reported in the form of mean and standard deviation over these 10 repetitions. The evaluation metrics for the experiments will encompass F1 score [Powers, 2020], Structural Hamming Distance (SHD), and Structural Intervention Distance (SID) [Peters and Bühlmann, 2015].

Specifically, precision denotes the proportion of predicted edges that are actually present among the true edges, while recall represents the proportion of true edges that have been correctly predicted. F1 score is the weighted harmonic mean of both Precision and Recall, and is calculated as $F1 = \frac{2 \times Precision \times Recall}{Precision + Recall}$, the higher F1 score the better ($\uparrow$). SHD indicates the number of edge insertions, deletions, or flips required to transform one graph into another, the lower SHD the better ($\downarrow$). SID is a measure that quantifies the similarity between two DAGs based on their corresponding causal inference statements, the lower SID the better ($\downarrow$).

In experiments, we use the following Granger causal discovery methods as comparisons: TNPAR [Liu et al., 2024], ADM4 [Zhou et al., 2013], CAUSE [Zhang et al., 2020], PCMCI [Runge, 2020], MLE_SGL [Xu et al., 2016] and ACD [Löwe et al., 2022]. We would like to point out that THP [Cai et al., 2024] and TransHP [Zuo et al., 2020] are not included because they are too inefficient to produce results within required time under most settings. All the experiments are conducted on a Linux server with two 3.00GHz Intel Xeon Gold 6354 CPUs and one RTX3090 GPU. All the models are implemented by PyTorch [Paszke et al., 2019], and the code is available at `https://github.com/MingjiaLi666/S2GCSL`.

### 4.1 Implementation and Hyperparameters

In the algorithm implementation of S$^2$GCSL, the involved hyperparameters include: the geodesic distance in the topological network $k$, the regularization coefficients $\lambda_1$, $\lambda_2$ and the pruning threshold $\rho$. In the synthetic experiments, we set these hyperparameters as follows: $k = 2$, $\rho = 1 \times 10^{-3}$. Worth noting that the proper values of $\lambda_1$ and $\lambda_2$ are related to the scale of sample size and thus vary over different settings. We adopt a heuristic way to estimate the proper range of $\lambda_1$, $\lambda_2$ that calculate the local gradient of loss over $||\boldsymbol{A}'||_1$ and $h(\boldsymbol{A}')$ respectively. In the real-world experiments, we set the hyperparameters as follows: $k = 1$, $\rho = 5 \times 10^{-4}$, $\lambda_1 = 5 \times 10^5$ and $\lambda_2 = 1 \times 10^5$.

### 4.2 Synthetic Experiments

#### 4.2.1 Synthetic Experimental Setup

The synthetic data is generated in the following manner: a) A Granger causal graph and a topological graph are randomly created using $Erd\ddot{o}s~R\acute{e}nyi$ random graph generator. b) Root event records are produced using a Poisson process with a base intensity parameter $\mu$ in the Hawkes process. These root event records occur spontaneously within the system. c) Based on the root event records, propagated event records are discretely generated according to both the time interval $\Delta$ and the excitation coefficient $\alpha$. It is worth noting that event sequences can be sparse in real-world scenarios. To address this, a time interval parameter $\Delta$ is introduced to the generation process, dividing the time domain $[0, T]$ into small intervals with indexes as $\{1, \dots, \lceil T/\Delta \rceil\}$ where $\lceil \cdot \rceil$ refers to the ceiling function. Then, the event records can be summarized within the same timestamp. Note that $\Delta \ge 0$, and $\Delta = 0$ implies the use of the original event sequences. The simulated data is generated by systematically varying one parameter at a time in the generation process, while keeping the default parameters constant. The parameters included in the generation

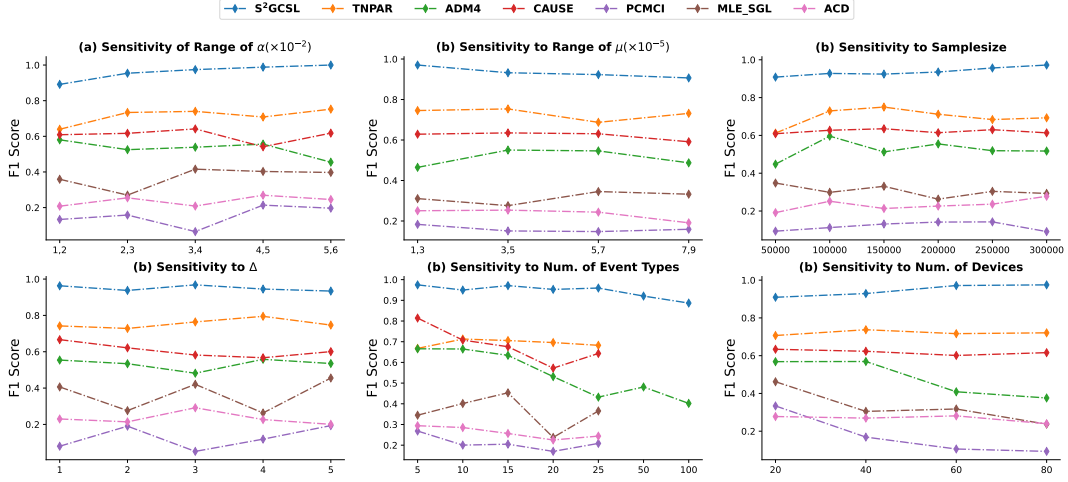

Figure 2: The F1 Scores of different methods on synthetic data.

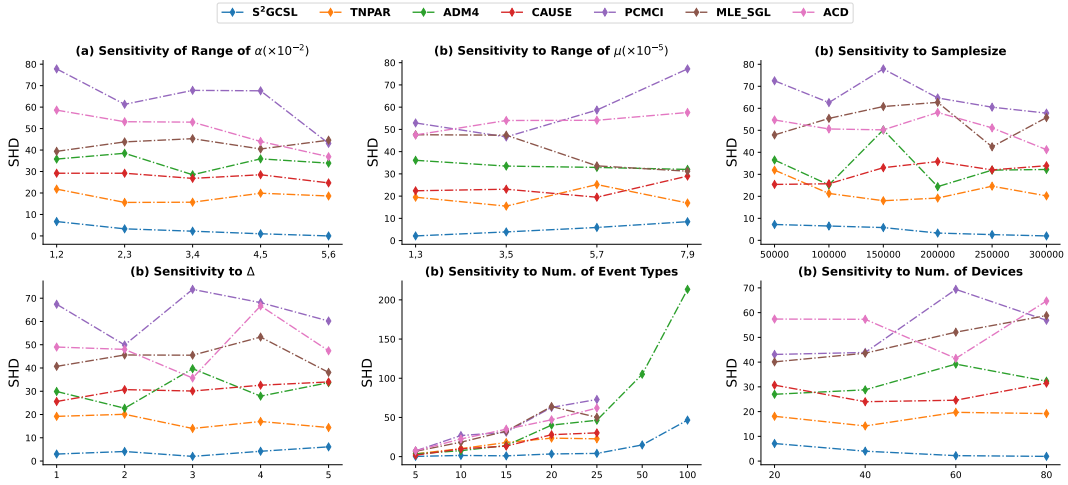

Figure 3: The SHD of different methods on synthetic data.

process are as follows: the number of devices ($|\boldsymbol{N}|$), the number of event types ($|\boldsymbol{V}|$), the sample size ($m$), the range of baseline intensity ($\mu$ range), the range of activation intensity ($\alpha$ range) and the time interval $\Delta$. The ranges of the above parameters are: $|\boldsymbol{N}| = \{20, \mathbf{40}, 60, 80\}$, $|\boldsymbol{V}| = \{5, 10, 15, \mathbf{20}, 25, 50, 100\}$, $m = \{50\,000, 100\,000, 150\,000, \mathbf{200\,000}, 250\,000, 300\,000\}$, $\mu$ range $(\times10^{-5}) = \{(1,3), \mathbf{(3,5)}, (5,7), (7,9)\}$, $\alpha$ range $(\times10^{-2}) = \{(1,2), \mathbf{(2,3)}, (3,4), (4,5), (5,6)\}$ and $\Delta = \{1, \mathbf{2}, 3, 4, 5\}$, where the default parameters are denoted in bold.

### 4.2.2 Results on Synthetic Data

**Effectiveness.** The F1 Score, SHD and SID of different methods on the synthetic data are shown in Figure 2, 3 and 4 respectively. Note that if the result for a particular baseline is missing at some points, it indicates that the method is unable to produce results within 1 hour (S$^2$GCSL gives results for problems with 100 event types in approximately 9 minutes.) under the setting of such points. Based on the results depicted in these 3 figures, S$^2$GCSL exhibit superior effectiveness across all scenarios when compared to other baselines. Besides, from Figure 2, S$^2$GCSL demonstrates strong robustness across all parameters including both baseline activation intensity, the sample size of data, the time interval of recording, the causal graph scale and the topological network scale. Such results indicate the robustness of S$^2$GCSL, i.e., S$^2$GCSL is expected to be effective in most scenarios.

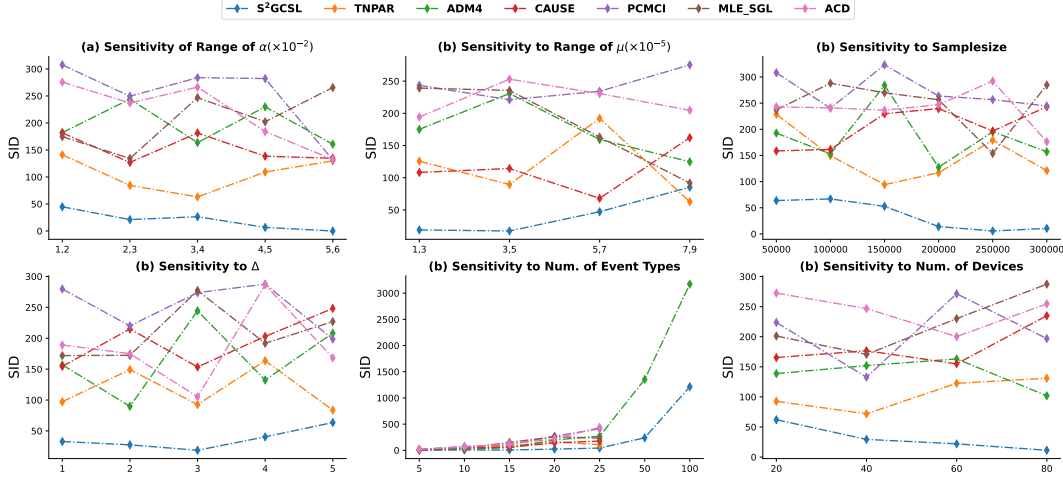

Figure 4: The SID of different methods on synthetic data.

Table 1: The wall-clock execution time (s) of different methods on different scale of synthetic problems. The algorithm with the highest efficiency under each scale of problem is marked in bold, and "-" indicates that results cannot be obtained within one hour.

| Algorithms | 5 | 10 | 15 | 20 | 25 | 50 | 100 |
|---|---|---|---|---|---|---|---|
| S$^2$GCSL | $\mathbf{2.48 \times 10^0}$ | $\mathbf{8.67 \times 10^0}$ | $\mathbf{1.92 \times 10^1}$ | $5.48 \times 10^1$ | $8.64 \times 10^1$ | $\mathbf{2.11 \times 10^2}$ | $\mathbf{5.42 \times 10^2}$ |
| TNPAR | $3.61 \times 10^2$ | $6.40 \times 10^2$ | $7.82 \times 10^2$ | $9.93 \times 10^2$ | $1.46 \times 10^3$ | - | - |
| ADM4 | $1.17 \times 10^1$ | $2.11 \times 10^1$ | $3.00 \times 10^1$ | $\mathbf{4.46 \times 10^1}$ | $\mathbf{6.68 \times 10^1}$ | $2.51 \times 10^2$ | $7.58 \times 10^2$ |
| CAUSE | $6.88 \times 10^2$ | $9.05 \times 10^2$ | $1.21 \times 10^3$ | $1.66 \times 10^3$ | $1.92 \times 10^3$ | - | - |
| PCMCI | $1.70 \times 10^1$ | $2.58 \times 10^2$ | $8.91 \times 10^2$ | $1.78 \times 10^3$ | $2.86 \times 10^3$ | - | - |
| MLE_SGL | $1.28 \times 10^2$ | $3.22 \times 10^2$ | $6.04 \times 10^2$ | $8.23 \times 10^2$ | $1.08 \times 10^3$ | - | - |
| ACD | $3.80 \times 10^1$ | $9.90 \times 10^1$ | $1.93 \times 10^2$ | $2.35 \times 10^2$ | $4.70 \times 10^2$ | - | - |

**Efficiency.** The wall-clock execution time results of different methods on different scale of problems are presented in Table 1. According to Table 1, S$^2$GCSL is able to produce results within 10 minutes even for problems scaling up to 100 event types. Worth noting that only two of the compared methods, namely S$^2$GCSL and ADM4, are scalable to problems with 50 event types, both of which have nearly an order of magnitude advantage on efficiency over other methods in comparison. Besides, compared to ADM4, S$^2$GCSL is more efficient on larger-scale problems (25 and 50), indicating its greater potential for application in large-scale real-world scenarios (Given that the real-world problem involved in this paper is with 38 event types).

**Summary.** To sum up, considering both effectiveness and efficiency, S$^2$GCSL demonstrates clear advantages in effectiveness and scalability compared to other methods, thanks to its simple yet problem-adapted design. In the next subsection, this paper will study the performance of S$^2$GCSL on a real-world TNFD task in metropolitan telecommunication networks.

### 4.3 Real-World Experiments

#### 4.3.1 Metropolitan Telecommunication Network Alarm Data

The real-world dataset utilized in this study was sourced from a business setting in metropolitan telecommunication network collected by a multinational communications company and is accessible through the NeurIPS 2023 competition of *Causal Structure Learning from Event Sequences and Prior Knowledge* [3] (Phase 2 real-world dataset). It comprises a sequence of alarm records generated based on both a topological network $\mathcal{G}_N$ and a causal structure $\mathcal{G}_V$. The topological network $\mathcal{G}_N$ consists of

Table 2: Performances of different methods on metropolitan telecommunication network alarm data. The algorithm perform best under each metric is highlighted in bold.

| Algorithms | F1 Score ($\uparrow$) | SHD ($\downarrow$) | SID ($\downarrow$) | ET(s) ($\downarrow$) |
|---|---|---|---|---|
| $S^2$GCSL | $\mathbf{0.40_{\pm 0.06}}$ | $\mathbf{60.6_{\pm 6.59}}$ | $397_{\pm 30.2}$ | $\mathbf{737}$s |
| TNPAR | $0.23_{\pm 0.06}$ | $83.1_{\pm 6.07}$ | $543_{\pm 62.6}$ | 4604s |
| ADM4 | $0.19_{\pm 0.03}$ | $83.5_{\pm 4.15}$ | $475_{\pm 50.4}$ | 861s |
| CAUSE | $0.29_{\pm 0.04}$ | $78.1_{\pm 4.09}$ | $468.7_{\pm 29.4}$ | 7209s |
| PCMCI | $0.08_{\pm 0.02}$ | $75.5_{\pm 4.32}$ | $\mathbf{367_{\pm 17.1}}$ | 9342s |
| MLE_SGL | $0.19_{\pm 0.05}$ | $77.2_{\pm 4.77}$ | $406_{\pm 29.0}$ | 3253s |
| ACD | $0.14_{\pm 0.04}$ | $107_{\pm 6.07}$ | $655_{\pm 54.6}$ | 1943s |

65 devices, and the alarm records cover 38 types of alarm events, amounting to 492,877 alarm event records. It is worth noting that, due to the equipment characteristics and as verified by experts, the collected timestamps of alarm events display specific time intervals.

#### 4.3.2 Results on Metropolitan Telecommunication Network Alarm Data

The results of the experiments on metropolitan telecommunication network alarm data can be found in Table 2, where the F1 Score, SHD, SID and wall clock execution time (ET(s)) for the compared methods are reported. The results show that the methods incorporating the topological network ($S^2$GCSL, CAUSE and TNPAR) outperform those that rely on the i.i.d. assumption with respect to F1 Score. This suggests that in the real-world telecommunications networks, alarm events do indeed propagate through devices within the topological network, consistent with the expert's knowledge and our model assumptions. However, both TNPAR and CAUSE take over an hour (for CAUSE, it needs 2 hours) to produce results, which is unacceptable in the race against time for TNFD, compared to just over ten minutes for $S^2$GCSL. From the perspective of effectiveness, $S^2$GCSL shows the highest F1 Score and the lowest SHD, indicating its relative insensitivity to weak Granger causal strength between event types, which may exhibit in real-world scenarios [Liu et al., 2024]. Additionally, $S^2$GCSL achieves a good SID, only being slightly higher than PCMCI (however, this is because PCMCI often deduce very sparse causal graphs), which highlight $S^2$GCSL's superior capability for causal inference [Peters and Bühlmann, 2015]. Therefore, from a comprehensive perspective, only $S^2$GCSL has the potential to be applied and work well in real-world scenarios.

## 5 Conclusion

This paper addresses the critical issue of fault diagnosis in large-scale telecommunication networks by proposing a simple method for efficient and scalable Granger causal structural learning from topological event sequences. The proposed approach, referred to as $S^2$GCSL, leverages a linear kernel to model the activation effects among different event types and uses gradient descent for the optimization of the likelihood function. This method simplifies the modeling and the optimization process to enhance the scalability and efficiency of inferring causal structures from large datasets of network alarms. Besides, $S^2$GCSL also include the integration of expert knowledge as constraints within the optimization process, ensuring the model's explainability via aligning with domain-specific insights, which is crucial for practical applications. By conducting extensive experiments on both synthetic datasets and real-world data from a metropolitan telecommunication network, we verify that $S^2$GCSL significantly outperforms existing methods in terms of scalability or efficiency, while maintaining good effectiveness.

Despite the advancements presented in this paper, there are several limitations that suggest avenues for future research. Firstly, while $S^2$GCSL provides an efficient solution for large-scale datasets, its application is currently limited by the simplicity of its modeling approach. The linear kernel may not capture the full complexity of interactions in more dynamically evolving network environments where non-linear relationships and non-stationary patterns prevail. Additionally, further research could focus on extending the applicability of $S^2$GCSL to other types of networks, such as power grids

or transport networks, where similar challenges in fault diagnosis are present. This would involve adapting the current framework to different kinds of event data and potentially different topologies.

## Acknowledgments

The authors would like to express the sincere thanks to the anonymous reviewers for their constructive and helpful comments. This work is supported by the National Natural Science Foundation of China (62476091, 62106076).

## Footnotes

[3]`https://codalab.lisn.upsaclay.fr/competitions/13902#learn_the_details-dataset`

## References

Jérémie Bouttier, Philippe Di Francesco, and Emmanuel Guitter. Geodesic distance in planar graphs. *Nuclear physics B*, 663(3):535–567, 2003.

Brillinger and David R. Time series, point processes, and hybrids. *Canadian Journal of Statistics*, 22 (2):177–206, 1994.

Ruichu Cai, Siyu Wu, Jie Qiao, Zhifeng Hao, Keli Zhang, and Xi Zhang. THPs: Topological hawkes processes for learning causal structure on event sequences. *IEEE Transactions on Neural Networks and Learning Systems*, 35(1):479–493, 2024.

Wei Chen, Ruichu Cai, Zhifeng Hao, Chang Yuan, and Feng Xie. Mining hidden non-redundant causal relationships in online social networks. *Neural Computing and Applications*, 32(11): 6913–6923, 2020.

Cox and David R. Some statistical methods connected with series of events. *Journal of the Royal Statistical Society: Series B (Methodological)*, 17(2):129–157, 1955.

Şeyda Ertekin, Cynthia Rudin, and Tyler H McCormick. Reactive point processes: A new approach to predicting power failures in underground electrical systems. *The Annals of Applied Statistics*, 9 (1):122–144, 2015.

Clive WJ Granger. Investigating causal relations by econometric models and cross-spectral methods. *Econometrica: Journal of the Econometric Society*, pages 424–438, 1969.

Hawkes and Alan G. Spectra of some self-exciting and mutually exciting point processes. *Biometrika*, 58(1):83–90, 1971.

Isham, Valerie, Westcott, and Mark. A self-correcting point process. *Stochastic Processes and Their Applications*, 8(3):335–347, 1979.

Diederik P. Kingma and Jimmy Ba. Adam: A method for stochastic optimization. In *Proceeding of the 3rd International Conference on Learning Representations*, San Diego, California, 2015.

Yuequn Liu, Ruichu Cai, Wei Chen, Jie Qiao, Yuguang Yan, Zijian Li, Keli Zhang, and Zhifeng Hao. Tnpar: Topological neural poisson auto-regressive model for learning granger causal structure from event sequences. In *Proceedings of the 38th AAAI Conference on Artificial Intelligence*, number 18, pages 20491–20499, 2024.

Sindy Löwe, David Madras, Richard S. Zemel, and Max Welling. Amortized causal discovery: Learning to infer causal graphs from time-series data. In *Proceedings of the 1st Conference on Causal Learning and Reasoning*, pages 509–525, Eureka, California, 2022.

Gorana Mijatovic, Yuri Antonacci, Tatjana Loncar-Turukalo, Ludovico Minati, and Luca Faes. An information-theoretic framework to measure the dynamic interaction between neural spike trains. *IEEE Transactions on Biomedical Engineering*, 68(12):3471–3481, 2021.

Adam Paszke, Sam Gross, Francisco Massa, Adam Lerer, James Bradbury, Gregory Chanan, Trevor Killeen, Zeming Lin, Natalia Gimelshein, Luca Antiga, Alban Desmaison, Andreas Köpf, Edward Z. Yang, Zachary DeVito, Martin Raison, Alykhan Tejani, Sasank Chilamkurthy, Benoit Steiner, Lu Fang, Junjie Bai, and Soumith Chintala. Pytorch: An imperative style, high-performance deep learning library. In *Advances in in the Annual Conference on Neural Information Processing Systems 32*, pages 8024–8035, British Columbia, Canada, 2019.

Jonas Peters and Peter Bühlmann. Structural intervention distance for evaluating causal graphs. *Neural Computation*, 27(3):771–799, 2015.

David M. W. Powers. Evaluation: from precision, recall and f-measure to roc, informedness, markedness and correlation. *CoRR*, abs/2010.16061, 2020.

Jakob Runge. Discovering contemporaneous and lagged causal relations in autocorrelated nonlinear time series datasets. In *Proceedings of the 36th Conference on Uncertainty in Artificial Intelligence*, pages 1388–1397, Virtual, 2020.

Oleksandr Shchur, Ali Caner Türkmen, Tim Januschowski, and Stephan Günnemann. Neural temporal point processes: A review. In *Proceedings of the 30th International Joint Conference on Artificial Intelligence*, pages 4585–4593, Montreal, Canada, 2021.

Hongteng Xu, Mehrdad Farajtabar, and Hongyuan Zha. Learning granger causality for hawkes processes. In *Proceedings of the 33rd International Conference on Machine Learning*, pages 1717–1726, New York City, New York, 2016.

Wei Zhang, Thomas Kobber Panum, Somesh Jha, Prasad Chalasani, and David Page. CAUSE: learning granger causality from event sequences using attribution methods. In *Proceedings of the 37th International Conference on Machine Learning*, pages 11235–11245, Virtual, 2020.

Xun Zheng, Bryon Aragam, Pradeep Ravikumar, and Eric P. Xing. Dags with NO TEARS: continuous optimization for structure learning. In *Advances in the Annual Conference on Neural Information Processing Systems 31*, pages 9492–9503, Montreal, Canada, 2018.

Ke Zhou, Hongyuan Zha, and Le Song. Learning social infectivity in sparse low-rank networks using multi-dimensional hawkes processes. In *Proceedings of the 16th International Conference on Artificial Intelligence and Statistics*, pages 641–649, Scottsdale, Arizona, 2013.

Simiao Zuo, Haoming Jiang, Zichong Li, Tuo Zhao, and Hongyuan Zha. Transformer hawkes process. In *Proceeding of the 37th International Conference on Machine Learning*, pages 11692–11702, Virtual, 2020.

